# Transient Signal Detection with Neural Networks: The Search for the Desired Signal

Jose C. Principe and Abir Zahalka

Computational NeuroEngineering Laboratory
Department of Electrical Engineering
University of Florida, CSE 447
Gainesville, FL 32611
principe@synapse.ee.ufl.edu

## Abstract

Matched filtering has been one of the most powerful techniques employed for transient detection. Here we will show that a dynamic neural network outperforms the conventional approach. When the artificial neural network (ANN) is trained with supervised learning schemes there is a need to supply the desired signal for all time, although we are only interested in detecting the transient. In this paper we also show the effects on the detection agreement of different strategies to construct the desired signal. The extension of the Bayes decision rule (0/1 desired signal), optimal in static classification, performs worse than desired signals constructed by random noise or prediction during the background.

## 1    INTRODUCTION

Detection of poorly defined waveshapes in a nonstationary high noise background is an important and difficult problem in signal processing. The matched filter is the optimal linear filter assuming stationary noise [Thomas, 1969]. The application area that we are going to discuss is epileptic spike detection in the electrocorticogram (ECoG), where the matched filtering produce poor results [Barlow and Dubinsky, 1976], [Pola and Romagnoli, 1979] due to the variability of the spike shape and the ever changing background (the brain electric activity). Recently artificial neural networks have been applied to spike detection [Eberhart et al, 1989], [Gabor and

Seydal, 1992], but static neural network architectures were chosen. Here a static multilayer perceptron (MLP) will be augmented with a short term memory mechanism to detect spikes. In the way we utilized the dynamic neural net, the *ANN can be thought of as an extension of the matched filter to nonlinear models*, which we will refer to as *a neural template matcher*. In our implementation, the ANN looks directly at the input signal with a time window larger than the longest spike for the sampling frequency utilized. The input layer of the dynamic network is a delay line, and the data is clocked one sample at a time through the network. The desired signal is "1" following the occurrence of a spike. With this strategy we teach the ANN to produce an output of "1" when a waveform similar to the spike is present within the time window. A spike will be recognized when the ANN output is above a given threshold.

Unlike the matched filter, the ANN does not require a single, explicit waveform for the template (due to the spike shape variability some form of averaging is needed to create the "average" spike shape which is normally a poor compromise). Rather, the ANN will learn the important features of the transient class by training on many sample spikes, refining its approximation with each presentation. Moreover, the ANN using the sigmoid nonlinearity will have to necessarily represent the background activity, since the discriminant function in pattern space is established from information pertaining to all input classes. Therefore, the nonstationary nature of the background can be accommodated during network training and we can expect that the performance of the neural template matcher will be improved with respect to the matched filter.

We will not address here the normalization of the ECoG, nor the issues associated with the learning criterion [Zahalka, 1992]. The purpose of this paper is to delve on the design of the desired signal, and quantify the effect on performance. What should be the shape of the desired sinal for transient detection, when on-line supervised learning is employed? In our approach we decided to construct a desired signal that exists for all time. We shall point out that the existence of a desired signal for every sample will simplify supervised learning, since in principle the conventional backpropagation algorithm [Rumelhart et al, 1986] can be utilized instead of the more time consuming backpropagation through time [Werbos, 1990] or real-time recurrent learning [Williams and Zipzer, 1989]. The simplified learning algorithm may very well be one of the factors which will make ANNs learn on-line as adaptive linear filters do. The main decision regarding the desired signal for spike detection is to decide the value of the desired signal during the background (which for spikes represent 99% of the time), since we decided already that it should be "1" following the spike. Similar problems have been found in speech recognition, when patterns that exist in time need to be learned [Unnikrishnan et al, 1991] [Watrous et al, 1990]. Here we will experimentally compare three desired signals (Figure 1):

Desired signal I- Extrapolating the Bayes rule for static patterns [Makhoul], we will create a target of zero during the background, and a value of 1 following the spike, with a duration equal to the amount of time the spike is in the input window.

Desired signal II- During the background, a random, uniformly distributed (between -0.5 and 0.5), zero mean target signal. Same target as above following the spike.

Desired signal III- During the background the network will be trained as a one step predictor. Same target as above following the spike.

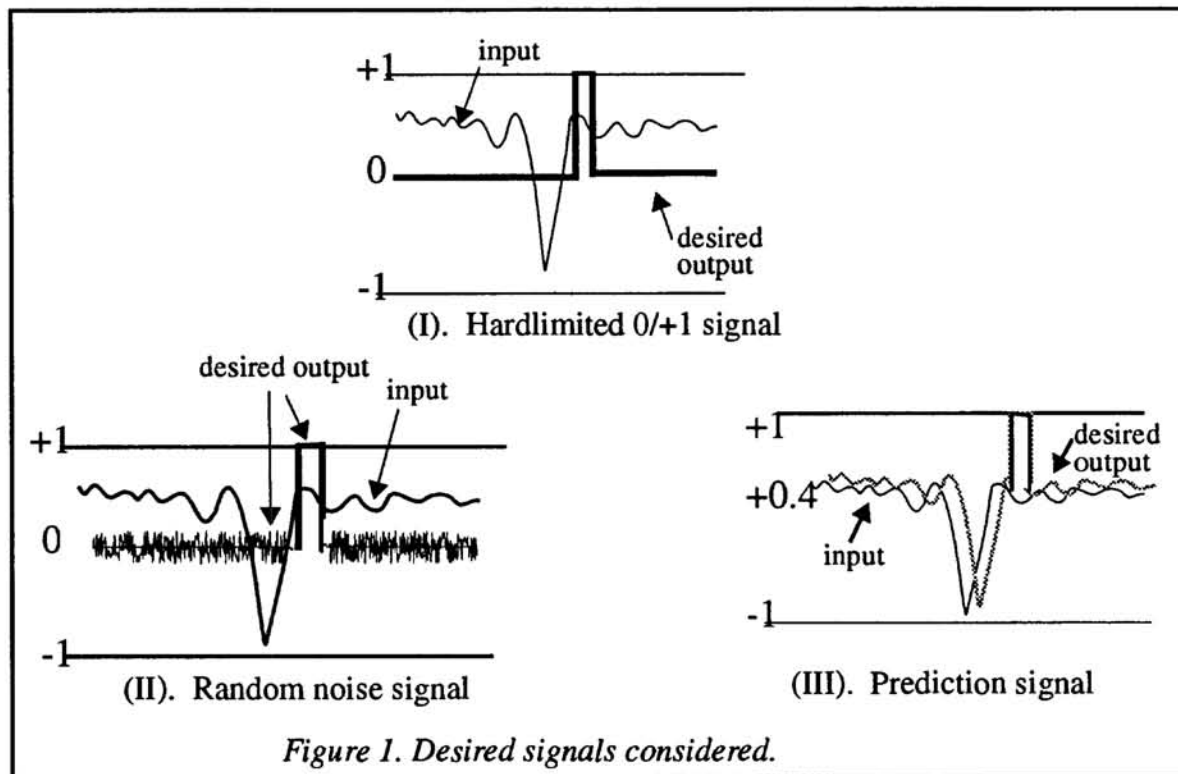

Figure 1. Desired signals considered.

## 2     NEURAL NETWORK ARCHITECTURE AND TRAINING

One ECoG channel was sampled at 500 Hz (12 bit A/D converter) and was pre-processed for normalization between -1 and 0.4, and DC removal before being presented to the ANN [Zahalka, 1992]. An epileptic spike has a duration between 20 and 70 msec.

The dynamic network used for this application is a time delay neural network (TDNN) consisting of an input layer with 45 taps, 12 hidden processing elements (PEs) and one linear output PE. The input window corresponds to 90 msec, so even the longest spike is fully present in the window during at least 10 samples.

The training set consists of 60 hand picked 2 sec. segments containing one spike each, embedded in 6,860 points of background activity. In principle the data could be streamed on-line to the ANN, provided that we could create also on-line the desired signal. But at this point of the research we preferred to control the segment length and choose well defined spikes to study training issues. The test data set consists of another set (belonging to the same individual) of 49 spikes embedded in 6,970 samples. A spike was defined when the ANN output was above 0.9.

The ANN was trained with the backpropagation algorithm [Rumelhart et al 1986]. The weights were updated after every sample (real-time mode). A momentum term ($\alpha$=0.9) was used, and 0.1 was added to the sigmoid derivative to speedup learning [Fahlman, 1988]. The training was stopped when the test set performance decreased. The network typically learned in less than 50 presentations of the training set. All the

results presented next use the same training/test sets, the same learning and stop criterion and the same network topology.

## 3    RESULTS

Desired Signal I.

We begin with the most commonly used desired signal for static classification, the hardlimited 0/+1 signal of Figure 1(I), but now extended in time (0 during the background, ı after the occurrence of the spike). This desired signal has been shown to be optimal in the sense that it is a least square approximation of the *Bayes decision rule [Makhoul, 1991]. However it performs poorly for transient detection* (73% of correct detections and 8% of false positives). We suspect that the problem lies in the different waveshapes present in the background. When an explicit 0 value is given as the desired signal for the background, the network has difficulties extracting common features to all the waves and biases the decision rule. Samples of the network input and the corresponding output are shown in Figure 2(I). Notice that the ANN output gets close to "1" during high amplitude background, and fails to be above 0.9 during small amplitude spikes. In Table 1, the test set performance is better than the training set due to the fact that the spikes chosen for training happen to include more difficult cases.

| Table 1. Performance Results. | | | | |
|---|---|---|---|---|
| Desired Signal | Training Set detections | Training Set false positives | Test Set detections | Test Set false positives |
| 0 <--> +1 | 38/60 = 63% | 2/38 = 5% | 36/49 = 73% | 3/36 = 8% |
| noise | 54/60 = 90% | 0/54 = 0% | 46/49 = 94% | 1/47 = 2% |
| prediction | 50/60 = 83% | 1/50 = 2% | 45/49 = 92% | 2/45 = 4% |

*Detections mean number of events in agreement between the human expert and the ANN (normalized by the number of spikes in the set).*

*False positives mean the number of events picked by the ANN, but not considered spikes by the human expert (normalized by the number of ANN detections)*

The "random noise" desired signal is shown in Figure 1(II). This signal consists simply of uniformly distributed random values bounded between -0.5 and +0.5 for the nonspike regions and again a value of "+1" after the spikes. This approach is based on the fact that the random number generator will have a zero mean distribution. Therefore, during training over the nonspike regions, the errors backpropagated through the network will normally average out to zero, yielding in practice a "don't care" target for the background. *This effectively should give the least bias to the decision rule.* The net result is that consistent training is only performed over the spike patterns, allowing the network to better learn these patterns and provide improved performance. The results of this configuration are very

promising, as can be seen in Table 1 and in Figure 2(II). The "noise" signal performs better than all of the other desired signal configurations examined (94% correct detections and 2% false detections).

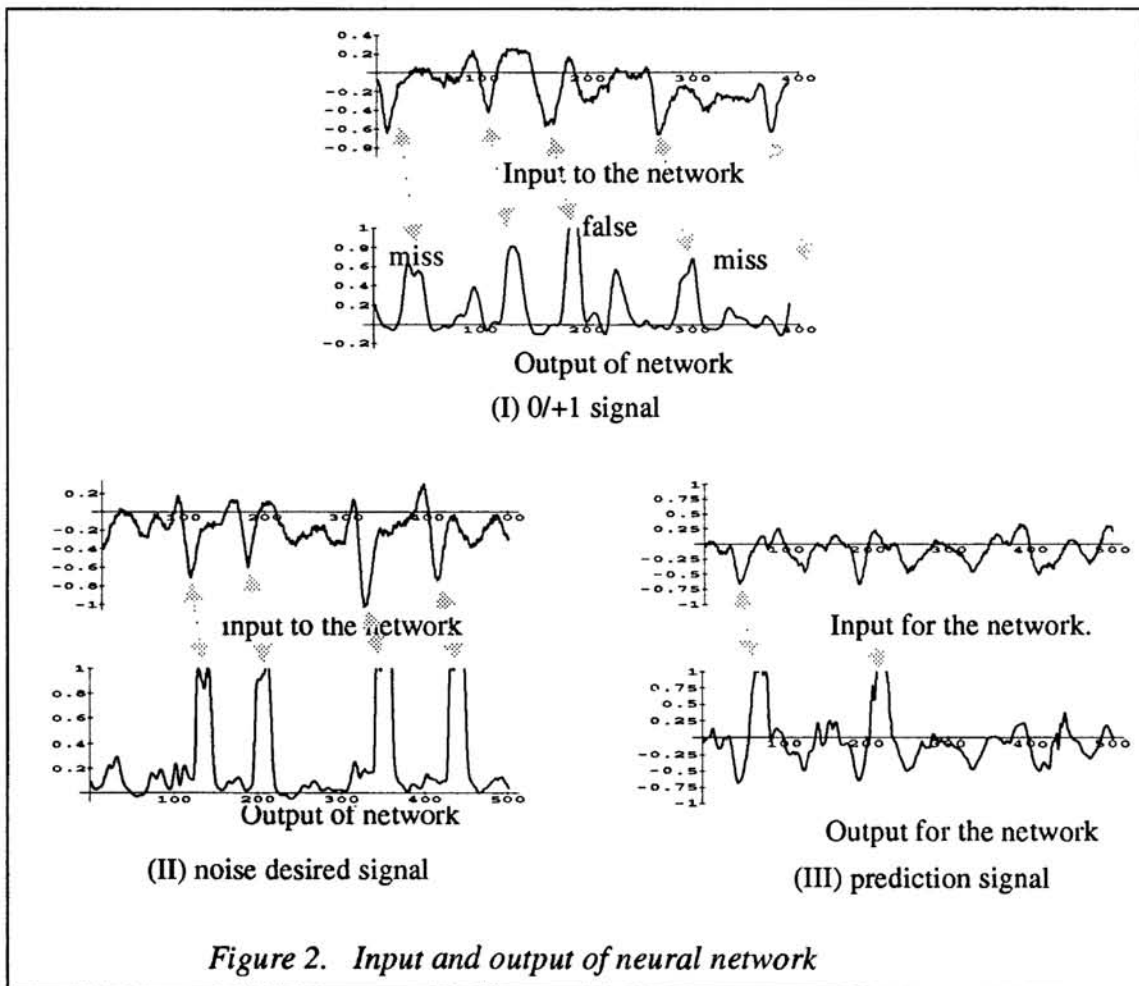

Figure 2.  Input and output of neural network

The last signal configuration is the prediction paradigm. The network is performing one-step prediction during the nonspike portions of the signal and a saturation value of "+1" is added to the desired signal after the spike, as in Figure 1(III). The rationale behind such a configuration is that the desired signal is readily available and may require fewer network resources than a target of "0", decreasing the bias in the decision rule for the spike class.   The results are given in Table 1, where we see a marked improvement over the hardlimited desired signal (92% correct detection and 4% false positives).

## 4    COMPARISON WITH MATCHED FILTER

The template for the matched filter was formed by averaging five spikes from the training set. While averaging reduces morphological "crispness," it enhances the robustness of spike shape representation, since the spike shape is variable. The 45 point template is correlated continuously with the time signal from the test set, hence

no correction to the nonwhite nature of the ECoG is being done. Performance in both the matched filter and the ANN approach will be dependent upon the threshold levels chosen for detection, which determines the receiver operating characteristic (ROC) for the detector, but requires large data sets to be meaningful. Table 2 shows a partial result, where both detectors are compared in terms of false positives for 90% detection rate and detection rate for 2 false positives. The results presented for the ANN are for the random noise desired signal. In both cases, we see the superiority of the ANN approach.

| Table 2. Comparison ANN/Matched filter | | | |
|---|---|---|---|
| 90% detection rate | | 2 false positives | |
| NN | MF | NN | MF |
| 0 false +'s | 21 false +'s | 92% | 13% |

## 5    CONCLUSIONS

*The ultimate conclusion of this experimental work is that neural networks can be used to implement transient detectors, outperforming the conventional matched filter for the detection of spike transients embedded in the ECoG.* However, the difficulty in the ANN approach comes in how to setup the training, and the desired signal.

Although the training was not discussed here, we would like to point several issues that are important. *When a cost function that is sensitive to the a priori probability of the classes is used (as the error signal in backpropagation) the proper balance of background versus spike data sizes is relevant for adequate training.* Our results show that at a low spike concentration of 4% (ratio of spike samples over background samples in the training set), the network never learns the waveform and always outputs the value chosen to represent the background. For our application, these problems disappear at an 18% concentration level.

Another important issue is the selection of the class exemplars for training. *Transient detection is a one class classification problem (A versus the universe)*, rather than a two class classification problem. It is not possible to cover appropriately the unconstrained background with examples, and even when a large number of background waves is utilized (in our case 80% of the training samples belonged to the background) the training produced bad results. We found out that only the waves similar in shape to the spikes are important to bound the spike class in feature space. As a solution, we *included in the training set the false positives*, i.e. waves that the system detects as spikes but the human expert classifies as background, to improve the detection agreement with the human [Zahalka, 1992].

The issues regarding the choice of the desired signal are also important. We found experimentally that the best desired signal for our case is the one that uses random noise during the background. We can not, at this point, explain this result

theoretically. This desired signal is also the one that uses fewer network resources (i. e. number of hidden units) without degrading the performance [Zahalka, 1992]. *This result came as a surprise since the 0/1 signal has been shown to be optimal for static patterns in the sense that makes the classifier approximate the Bayes decision rule.* The explanation may be simply a matter of local minima in the performance surface of the network trained with the 0/1 desired signal, but it may also reflect deeper causes. With the 0/1 desired signal, the network weights are updated equally with the information of the background waveforms and with the information regarding the spikes. The training paradigm should emphasize the spikes, since the spikes are the waves we are interested in. Moreover, since the background class is unconstrained, many more degrees of freedom are necessary to represent well the background. When not enough hidden units are available, the network biases the discriminant function, and the performance is poor. In the prediction paradigm the background is selected as the desired signal and the required number of hidden nodes to predict the next sample is much smaller (actually only three hidden nodes are sufficient to keep the reported performance [Zahalka, 1992]). The network resources naturally self organize as two template matching nodes and one prediction node. However, we have found out that the signal has to be properly normalized in the sense that the target for the spike ("1") must be outside the range of the input signal voltages (that is the reason we normalize the input signal between -1 and 0.4). From a classification point of view we "do not care" what the net output is provided it is far from the target of "1". The random noise target with zero mean achieves this goal very easily, because the error gradient during the background averages out to zero. Therefore the weights reflect primarily the information containing in the shape of the spike. We found out that only two hidden nodes are sufficient to keep the reported performance. [Zahalka, 1992]. It seems that the theory of time varying signal classification with neural networks is not a straight forward extension of the static classification case, and requires further theoretical analysis.

An implication of this work regards the role of supervised learning in biological neural networks. This work shows that during non-interesting events, there is no need to provide a target to the neural assembly (noise, which is so readily available in biological systems, suffices). Re-enforcement stimulus are only needed during or after relevant events, which is compatible with the information processing models of the olfactory bulb [Freeman and DiPrisco, 1989]. Therefore, looking at learning and adaptation in biological systems as a signal detection instead of a classification problem seems promising.

## Acknowledgments

This work has been partially supported by NSF grants ECS-9208789 and DDM-8914084.

## References

Barlow J. S. and Dubinsky J., (1976) "Some Computer Approaches to Continuous

Automatic Clinical EEG Monitoring," in *Quantitative Analytic Studies in Epilepsy*, Raven Press, New York, 309-327.

Eberhart R., Dobbins R., Weber W., (1989) "Casenet: a neural network tool for EEG waveform classifiaction", *Proc IEEE Symp. Comp. Based Medical Systems*, Minneapolis, 60-68, 1989.

Fahlman S.,(1988) "Faster learning variations on backpropagation: an empirical study", *Proc. 1988 Connectionist Summer School*, Morgan Kaufmann, 38-51.

Freeman W., DiPrisco V., (1986) "EEG spatial pattern differences with discriminated odors manifest chaotic and limit cycle attractors in olfactory bulb of rabbits", in *Brain Theory*, Ed. Palm and Aertsen, Springer, 97-120.

Gabor A., Seydal M., (1992) "Automated interictal EEG spike detection using artifical neural networks", *Electroenc. Clin. Neurophysiol.*, (83), 271-280.

Makhoul J., (1991) "Pattern recognition properties of neural networks",*Proc. 1991 IEEE Workshop Neural Net. in Sig. Proc.*, 173-187, Princeton.

Pola P. and Romagnoly O., (1979) "Automatic analysis of interictal epileptic activity related to its morphological aspects", *Electroenceph. Clin. Neurophysiol.*, #46, 227-231.

Rumelhart,D.E., Hinton,G.E. and Williams,R.J. (1986) "Learning internal representations by error propagation. in *Parallel Distributed Processing* (Rumelhart, McClelland, eds.), ch. 8, Cambridge, MA.

Thomas J., (1969) *"An Introduction to statistical Communication Theory"*, Wiley.

Watrous R., Ladendorf B., Kuhn G., (1990) "Complete gradient optimization of a recurrent network applied to b,d,g discrimination", *J. Acoust. Soc. Am.* 87 (3), 1301-1309.

Werbos, P.J. (1990) "Backpropagation through time: what it does and how to do it", *Proc. IEEE*, vol 78, no10, 1550-1560.

Williams,R.J. and Zipser, D. (1989) "A learning algorithm for continually running fully recurrent neural networks. in *Neural Computation, vol. 1 (2)*.

Unikrishnan K., Hopfield J., Tank D., (1991) "Connected-Digit Speaker-dependent speech recognition using a neural network with time delayed connections", *IEEE Trans. Sig Proc.*, vol 39, #3, 698-713.

Zalahka A., (1992) "Signal detection with neural networks: an application to the recognition of epileptic spikes", *Master Thesis*, University of FLorida.
